# Refining PID Controllers using Neural Networks

**Gary M. Scott**
Department of Chemical Engineering
1415 Johnson Drive
University of Wisconsin
Madison, WI 53706

**Jude W. Shavlik**
Department of Computer Sciences
1210 W. Dayton Street
University of Wisconsin
Madison, WI 53706

**W. Harmon Ray**
Department of Chemical Engineering
1415 Johnson Drive
University of Wisconsin
Madison, WI 53706

## Abstract

The KBANN approach uses neural networks to refine knowledge that can be written in the form of simple propositional rules. We extend this idea further by presenting the MANNCON algorithm by which the mathematical equations governing a PID controller determine the topology and initial weights of a network, which is further trained using backpropagation. We apply this method to the task of controlling the outflow and temperature of a water tank, producing statistically-significant gains in accuracy over both a standard neural network approach and a non-learning PID controller. Furthermore, using the PID knowledge to initialize the weights of the network produces statistically less variation in testset accuracy when compared to networks initialized with small random numbers.

## 1  INTRODUCTION

Research into the design of neural networks for process control has largely ignored existing knowledge about the task at hand. One form this knowledge (often called the "domain theory") can take is embodied in traditional controller paradigms. The

recently-developed KBANN (Knowledge-Based Artificial Neural Networks) approach (Towell et al., 1990) addresses this issue for tasks for which a domain theory (written using simple propositional rules) is available. The basis of this approach is to use the existing knowledge to determine an appropriate network topology and initial weights, such that the network begins its learning process at a "good" starting point.

This paper describes the MANNCON (Multivariable Artificial Neural Network Control) algorithm, a method of using a traditional controller paradigm to determine the topology and initial weights of a network. The used of a PID controller in this way eliminates network-design problems such as the choice of network topology (*i.e.*, the number of hidden units) and reduces the sensitivity of the network to the initial values of the weights. Furthermore, the initial configuration of the network is closer to its final state than it would normally be in a randomly-configured network. Thus, the MANNCON networks perform better and more consistently than the standard, randomly-initialized three-layer approach.

The task we examine here is learning to control a Multiple-Input, Multiple-Output (MIMO) system. There are a number of reasons to investigate this task using neural networks. One, it usually involves nonlinear input-output relationships, which matches the nonlinear nature of neural networks. Two, there have been a number of successful applications of neural networks to this task (Bhat & McAvoy, 1990; Jordan & Jacobs, 1990; Miller et al., 1990). Finally, there are a number of existing controller paradigms which can be used to determine the topology and the initial weights of the network.

## 2   CONTROLLER NETWORKS

The MANNCON algorithm uses a *Proportional–Integral–Derivative* (PID) controller (Stephanopoulos, 1984), one of the simplest of the traditional feedback controller schemes, as the basis for the construction and initialization of a neural network controller. The basic idea of PID control is that the control action $\mathbf{u}$ (a vector) should be proportional to the error, the integral of the error over time, and the temporal derivative of the error. Several tuning parameters determine the contribution of these various components. Figure 1 depicts the resulting network topology based on the PID controller paradigm. The first layer of the network, that from $\mathbf{y}_{sp}$ (desired process output or setpoint) and $\mathbf{y}_{(n-1)}$ (actual process output of the past time step), calculates the simple error ($\mathbf{e}$). A simple vector difference,

$$\mathbf{e} = \mathbf{y}_{sp} - \mathbf{y}$$

accomplishes this. The second layer, that between $\mathbf{e}$, $\boldsymbol{\varepsilon}_{(n-1)}$, and $\boldsymbol{\varepsilon}$, calculates the actual error to be passed to the PID mechanism. In effect, this layer acts as a steady-state pre-compensator (Ray, 1981), where

$$\boldsymbol{\varepsilon} = \mathbf{G_I}\mathbf{e}$$

and produces the current error and the error signals at the past two time steps. This compensator is a constant matrix, $\mathbf{G_I}$, with values such that interactions at a steady state between the various control loops are eliminated. The final layer, that between $\boldsymbol{\varepsilon}$ and $\mathbf{u}_{(n)}$ (controller output/plant input), calculates the controller action

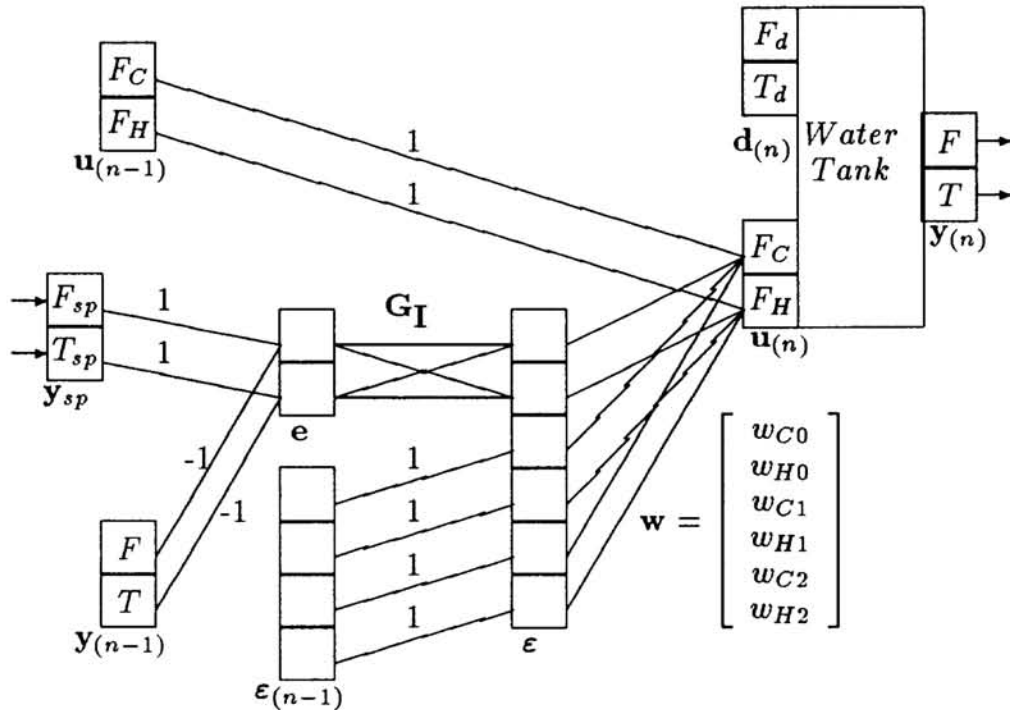

Figure 1: MANNCON network showing weights that are initialized using Ziegler-Nichols tuning parameters.

based on the velocity form of the discrete PID controller:

$$u_{C(n)} = u_{C(n-1)} + w_{C0}\varepsilon_{1(n)} + w_{C1}\varepsilon_{1(n-1)} + w_{C2}\varepsilon_{1(n-2)}$$

where $w_{C0}$, $w_{C1}$, and $w_{C2}$ are constants determined by the tuning parameters of the controller for that loop. A similar set of equations and constants $(w_{H0}, w_{H1}, w_{H2})$ exist for the other controller loop.

Figure 2 shows a schematic of the water tank (Ray, 1981) that the network controls. This figure also shows the controller variables ($F_C$ and $F_H$), the tank output variables ($F(h)$ and $T$), and the disturbance variables ($F_d$ and $T_d$). The controller cannot measure the disturbances, which represent noise in the system.

MANNCON initializes the weights of Figure 1's network with values that mimic the behavior of a PID controller tuned with Ziegler-Nichols (Z-N) parameters (Stephanopoulos, 1984) at a particular operating condition. Using the KBANN approach (Towell et al., 1990), it adds weights to the network such that all units in a layer are connected to all units in all subsequent layers, and initializes these weights to small random numbers several orders of magnitude smaller than the weights determined by the PID parameters. We scaled the inputs and outputs of the network to be in the range $[0, 1]$.

Initializing the weights of the network in the manner given above assumes that the activation functions of the units in the network are linear, that is,

$$o_{j,linear} = \sum w_{ji}o_i$$

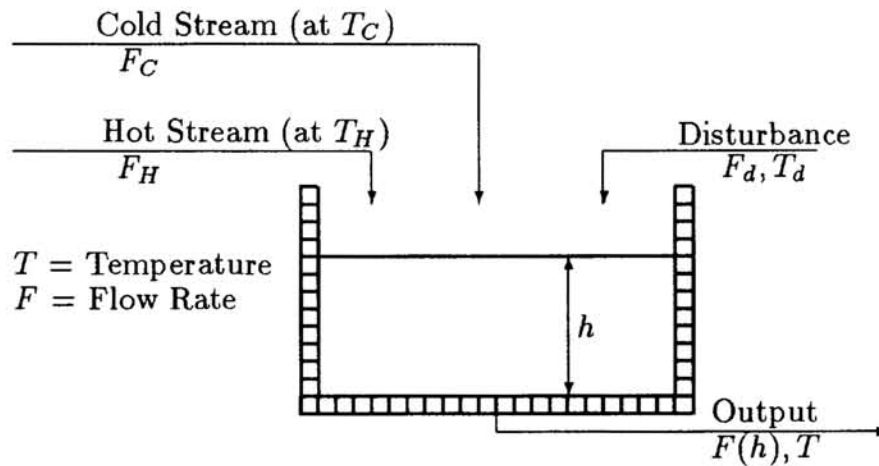

Figure 2: Stirred mixing tank requiring outflow and temperature control.

Table 1: Topology and initialization of networks.

| Network | Topology | Weight Initialization |
|---|---|---|
| 1. Standard neural network | 3-layer (14 hidden units) | random |
| 2. MANNCON network I | PID topology | random |
| 3. MANNCON network II | PID topology | Z-N tuning |

The strength of neural networks, however, lie in their having nonlinear (typically sigmoidal) activation functions. For this reason, the MANNCON system initially sets the weights (and the biases of the units) so that the linear response dictated by the PID initialization is *approximated* by a sigmoid over the output range of the unit. For units that have outputs in the range $[-1, 1]$, the activation function becomes

$$o_{j,sigmoid} = \frac{2}{1 + \exp(-2.31 \sum w_{ji}o_i)} - 1$$

where $w_{ji}$ are the linear weights described above.

Once MANNCON configures and initializes the weights of the network, it uses a set of training examples and backpropagation to improve the accuracy of the network. The weights initialized with PID information, as well as those initialized with small random numbers, change during backpropagation training.

## 3   EXPERIMENTAL DETAILS

We compared the performance of three networks that differed in their topology and/or their method of initialization. Table 1 summarizes the network topology and weight initialization method for each network. In this table, "PID topology" is the network structure shown in Figure 1. "Random" weight initialization sets

Table 2: Range and average duration of setpoints for experiments.

| Experiment | Training Set | Testing Set |
|:---:|:---:|:---:|
| 1 | $[0.1, 0.9]$ | $[0.1, 0.9]$ |
| | 22 instances | 22 instances |
| 2 | $[0.1, 0.9]$ | $[0.1, 0.9]$ |
| | 22 instances | 80 instances |
| 3 | $[0.4, 0.6]$ | $[0.1, 0.9]$ |
| | 22 instances | 80 instances |

all weights to small random numbers centered around zero. We also compare these networks to a (non-learning) PID controller.

We trained the networks using backpropagation over a randomly-determined schedule of setpoint $\mathbf{y}_{sp}$ and disturbance $\mathbf{d}$ changes that did not repeat. The setpoints, which represent the desired output values that the controller is to maintain, are the temperature and outflow of the tank. The disturbances, which represent noise, are the inflow rate and temperature of a disturbance stream. The magnitudes of the setpoints and the disturbances formed a Gaussian distribution centered at 0.5. The number of training examples between changes in the setpoints and disturbances were exponentially distributed.

We performed three experiments in which the characteristics of the training and/or testing set differed. Table 2 summarizes the range of the setpoints as well as their average duration for each data set in the experiments. As can be seen, in Experiment 1, the training set and testing sets were qualitatively similar; in Experiment 2, the test set was of longer duration setpoints; and in Experiment 3, the training set was restricted to a subrange of the testing set. We periodically interrupted training and tested the network. Results are averaged over 10 runs (Scott, 1991).

We used the error at the output of the tank ($\mathbf{y}$ in Figure 1) to determine the network error (at $\mathbf{u}$) by propagating the error backward through the plant (Psaltis et al., 1988). In this method, the error signal at the input to the tank is given by

$$\delta_{ui} = f'(net_{ui}) \sum_j \delta_{yj} \frac{\partial y_i}{\partial u_i}$$

where $\delta_{yj}$ represents the simple error at the output of the water tank and $\delta_{ui}$ is the error signal at the input of the tank. Since we used a *model* of the process and not a real tank, we can calculate the partial derivatives from the process model equations.

## 4    RESULTS

Figure 3 compares the performance of the three networks for Experiment 1. As can be seen, the MANNCON networks show an increase in correctness over the standard neural network approach. Statistical analysis of the errors using a *t*-test show that they differ significantly at the 99.5% confidence level. Furthermore, while the difference in performance between MANNCON network I and MANNCON network II is

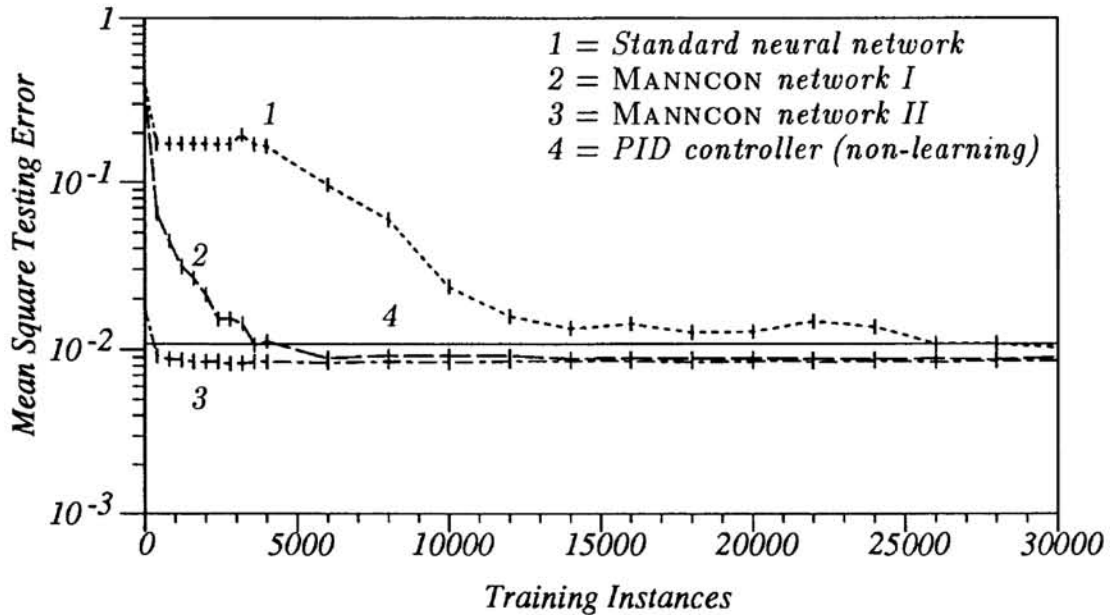

Figure 3: Mean square error of networks on the testset as a function of
the number of training instances presented for Experiment 1.

not significant, the difference in the *variance* of the testing error over different runs
is significant (99.5% confidence level). Finally, the MANNCON networks perform
significantly better (99.95% confidence level) than the non-learning PID controller.
The performance of the standard neural network represents the best of several trials
with a varying number of hidden units ranging from 2 to 20.

A second observation from Figure 3 is that the MANNCON networks learned much
more quickly than the standard neural-network approach. The MANNCON networks
required significantly fewer training instances to reach a performance level within
5% of its final error rate. For each of the experiments, Table 3 summarizes the
final mean error, as well as the number of training instances required to achieve a
performance within 5% of this value.

In Experiments 2 and 3 we again see a significant gain in correctness of the MAN-
NCON networks over both the standard neural network approach (99.95% confidence
level) as well as the non-learning PID controller (99.95% confidence level). In these
experiments, the MANNCON network initialized with Z-N tuning also learned sig-
nificantly quicker (99.95% confidence level) than the standard neural network.

## 5    FUTURE WORK

One question is whether the introduction of extra hidden units into the network
would improve the performance by giving the network "room" to learn concepts
that are outside the given domain theory. The addition of extra hidden units as
well as the removal of unneeded units is an area with much ongoing research.

Table 3: Comparison of network performance.

| Method | Mean Square Error | Training Instances |
|---|---|---|
| Experiment 1 | | |
| 1. Standard neural network | $0.0103 \pm 0.0004$ | $25,200 \pm 2,260$ |
| 2. MANNCON network I | $0.0090 \pm 0.0006$ | $5,000 \pm 3,340$ |
| 3. MANNCON network II | $0.0086 \pm 0.0001$ | $640 \pm 200$ |
| 4. PID control (Z-N tuning) | $0.0109$ | |
| 5. Fixed control action | $0.0190$ | |
| Experiment 2 | | |
| 1. Standard neural network | $0.0118 \pm 0.00158$ | $14,400 \pm 3,150$ |
| 2. MANNCON network I | $0.0040 \pm 0.00014$ | $12,000 \pm 3,690$ |
| 3. MANNCON network II | $0.0038 \pm 0.00006$ | $2,080 \pm 300$ |
| 4. PID control (Z-N tuning) | $0.0045$ | |
| 5. Fixed control action | $0.0181$ | |
| Experiment 3 | | |
| 1. Standard neural network | $0.0112 \pm 0.00013$ | $25,200 \pm 2,360$ |
| 2. MANNCON network I | $0.0039 \pm 0.00008$ | $25,000 \pm 1,550$ |
| 3. MANNCON network II | $0.0036 \pm 0.00006$ | $9,400 \pm 1,180$ |
| 4. PID control (Z-N tuning) | $0.0045$ | |
| 5. Fixed control action | $0.0181$ | |

The "$\pm$" indicates that the true value lies within these bounds at a 95% confidence level. The values given for fixed control action (5) represent the errors resulting from fixing the control actions at a level that produces outputs of $[0.5, 0.5]$ at steady state.

"Ringing" (rapid changes in controller actions) occurred in some of the trained networks. A future enhancement of this approach would be to create a network architecture that prevented this ringing, perhaps by limiting the changes in the controller actions to some relatively small values.

Another important goal of this approach is the application of it to other real-world processes. The water tank in this project, while illustrative of the approach, was quite simple. Much more difficult problems (such as those containing significant time delays) exist and should be explored.

There are several other controller paradigms that could be used as a basis for network construction and initialization. There are several different digital controllers, such as Deadbeat or Dahlin's (Stephanopoulos, 1984), that could be used in place of the digital PID controller used in this project. Dynamic Matrix Control (DMC) (Pratt et al., 1980) and Internal Model Control (IMC) (Garcia & Morari, 1982) are also candidates for consideration for this approach.

Finally, neural networks are generally considered to be "black boxes," in that their inner workings are completely uninterpretable. Since the neural networks in this approach are initialized with information, it may be possible to interpret the weights of the network and extract useful information from the trained network.

## 6   CONCLUSIONS

We have described the MANNCON algorithm, which uses the information from a PID controller to determine a relevant network topology without resorting to trial-and-error methods. In addition, the algorithm, through initialization of the weights with prior knowledge, gives the backpropagtion algorithm an appropriate direction in which to continue learning. Finally, we have shown that using the MANNCON algorithm significantly improves the performance of the trained network in the following ways:

- Improved mean testset accuracy
- Less variability between runs
- Faster rate of learning
- Better generalization and extrapolation ability

**Acknowledgements**

This material based upon work partially supported under a National Science Foundation Graduate Fellowship (to Scott), Office of Naval Research Grant N00014-90-J-1941, and National Science Foundation Grants IRI-9002413 and CPT-8715051.

**References**

Bhat, N. & McAvoy, T. J. (1990).  Use of neural nets for dynamic modeling and control of chemical process systems. *Computers and Chemical Engineering, 14,* 573–583.

Garcia, C. E. & Morari, M. (1982).  Internal model control: 1. A unifying review and some new results. *I&EC Process Design & Development, 21,* 308–323.

Jordan, M. I. & Jacobs, R. A. (1990).  Learning to control an unstable system with forward modeling. In *Advances in Neural Information Processing Systems* (Vol. 2, pp. 325–331). San Mateo, CA: Morgan Kaufmann.

Miller, W. T., Sutton, R. S., & Werbos, P. J. (Eds.)(1990).  *Neural networks for control.* Cambridge, MA: MIT Press.

Pratt, D. M., Ramaker, B. L., & Cutler, C. R. (1980).  Dynamic matrix control method. Patent 4,349,869, Shell Oil Company.

Psaltis, D., Sideris, A., & Yamamura, A. A. (1988).  A multilayered neural network controller. *IEEE Control Systems Magazine, 8,* 17–21.

Ray, W. H. (1981).  *Advanced process control.* New York: McGraw-Hill, Inc.

Scott, G. M. (1991).  Refining PID controllers using neural networks.  Master's project, University of Wisconsin, Department of Computer Sciences.

Stephanopoulos, G. (1984).  *Chemical process control: An introduction to theory and practice.* Englewood Cliffs, NJ: Prentice Hall, Inc.

Towell, G., Shavlik, J., & Noordewier, M. (1990).  Refinement of approximate domain theories by knowledge-base neural networks. In *Eighth National Conference on Aritificial Intelligence* (pp. 861–866). Menlo Park, CA: AAAI Press.